# A Hierarchical Bayesian Markovian Model for Motifs in Biopolymer Sequences

**Eric P. Xing, Michael I. Jordan, Richard M. Karp and Stuart Russell**
Computer Science Division
University of California, Berkeley
Berkeley, CA 94720
{epxing,jordan,karp,russell}@cs.berkeley.edu

## Abstract

We propose a dynamic Bayesian model for motifs in biopolymer sequences which captures rich biological prior knowledge and positional dependencies in motif structure in a principled way. Our model posits that the position-specific multinomial parameters for monomer distribution are distributed as a latent Dirichlet-mixture random variable, and the position-specific Dirichlet component is determined by a hidden Markov process. Model parameters can be fit on training motifs using a variational EM algorithm within an empirical Bayesian framework. Variational inference is also used for detecting hidden motifs. Our model improves over previous models that ignore biological priors and positional dependence. It has much higher sensitivity to motifs during detection and a notable ability to distinguish genuine motifs from false recurring patterns.

## 1 Introduction

The identification of motif structures in biopolymer sequences such as proteins and DNA is an important task in computational biology and is essential in advancing our knowledge about biological systems. For example, the gene regulatory motifs in DNA provide key clues about the regulatory network underlying the complex control and coordination of gene expression in response to physiological or environmental changes in living cells [11].

There have been several lines of research on statistical modeling of motifs [7, 10], which have led to algorithms for motif detection such as MEME [1] and BioProspector [9] Unfortunately, although these algorithms work well for simple motif patterns, often they are incapable of distinguishing what biologists would recognize as a true motif from a random recurring pattern [4], and provide no mechanism for incorporating biological knowledge of motif structure and sequence composition.

Most motif models assume independence of position-specific multinomial distributions of monomers such as nucleotides (nt) and amino acids (aa). Such strategies contradict our intuition that the sites in motifs naturally possess spatial dependencies for functional reasons. Furthermore, the vague Dirichlet prior used in some of these models acts as no more than a smoother, taking little consideration of the rich prior knowledge in biologically identified motifs. In this paper we describe a new model for monomer distribution in motifs. Our model is based on a finite set of informative Dirichlet distributions and a (first-order) Markov model for transitions between Dirichlets. The distribution of the monomers is

a continuous mixture of position-specific multinomials which admit a Dirichlet prior according to the hidden Markov states, introducing both multi-modal prior information and dependencies. We also propose a framework for decomposing the general motif model into a local alignment model for motif pattern and a global model for motif instance distribution, which allows complex models to be developed in a modular way.

To simplify our discussion, we use DNA motif modeling as a running example in this paper, though it should be clear that the model is applicable to other sequence modeling problems.

## 2 Preliminaries

DNA motifs are short (about 6-30 bp) stochastic string patterns (Figure 1) in the regulatory sequences of genes that facilitate control functions by interacting with specific transcriptional regulatory proteins. Each motif typically appears once or multiple times in the control regions of a small set of genes. Each gene usually harbors several motifs. We do not know the patterns of most motifs, in which gene they appear and where they appear. The goal of motif detection is to identify instances of possible motifs hidden in sequences and learn a model for each motif for future prediction.

A regulatory DNA sequence can be fully specified by a character string $y = (y_1, \ldots, y_T) \in \{A,T,C,G\}^T$, and an indicator string $x$ that signals the locations of the motif occurrences. The reason to call a motif a stochastic string pattern rather than a word is due to the variability in the "spellings" of different instances of the same motif in the genome. Conventionally, biologists display a motif pattern (of length $L$) by a *multi-alignment* $\mathbf{A}$ of all its $M$ instances. The stochasticity of motif patterns is reflected in the heterogeneity of nucleotide species appearing in each *column* (corresponding to a *position* or *site* in the motif) of the multi-alignment. We denote the multi-alignment of all instances of a motif specified by the indicator string $x$ in sequence $y$ by $\mathbf{A}(x, y)$. Since any $\mathbf{A}(x, y)$ can be characterized by the nucleotide counts for each column, we define a *counting matrix* $h(\mathbf{A})$ (or $h(x, y)$), where each column $\vec{h}_l = \langle h_{l1}, \ldots, h_{l4} \rangle$ is an integer vector with four elements, giving the number of occurrences of each nucleotide at position $l$ of the motif. (Similarly we can define the *counting vector* $\vec{h}_0$ for the whole sequence $y$.) With these settings, one can model the nt-distribution of a position $l$ of the motif by a *position-specific multinomial distribution*, $\theta_l = \langle \theta_{l1}, \ldots, \theta_{l4} \rangle$. Formally, the problem of inferring $X = \{x^{(1)}, \ldots, x^{(N)}\}$ and $\Theta = \{\theta_1, \ldots, \theta_L\}$ (often called a position-weight matrix, or PWM), given a sequence set $Y = \{y^{(1)}, \ldots, y^{(N)}\}$, is motif detection in a nutshell [1].

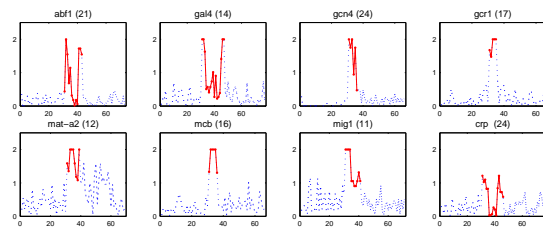
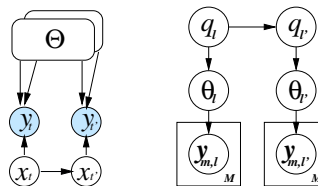

Figure 1: Yeast motifs (solid line) with $\pm$ 30 bp flanking regions (dashed line). The $x$ axis indexes position and the $y$ axis represents the information content $2 - H(\theta_l)$ of the multinomial distribution $\theta_l$ of nt at position $l$. Note the two typical patterns: the *U*-shape and the *bell*-shape.

Figure 2: (Left) A general motif model is a Bayes-ian multinet. Conditional on the value of $x$, $y$ admits different distributions (round-cornered boxes) parameterized by $\Theta$. (Right) The HMDM model for motif instances specified by a given $x$. Boxes are plates representing replicates.

# 3 Generative models for regulatory DNA sequences

## 3.1 General setting and related work

Without loss of generality, assume that the occurrences of motifs in a DNA sequence, as indicated by $x$, are governed by a global distribution $p(x|\Theta_g, \mathcal{M}_g)$; for each type of motif, the nucleotide sequence pattern shared by all its instances admits a local alignment model $p(\mathbf{A}(x,y)|x, \Theta_l, \mathcal{M}_l)$. (Usually, the background non-motif sequences are modeled by a simple conditional model, $p(y - \mathbf{A}(y,x)|x, \Theta_{bk})$, where the background nt-distribution parameters $\Theta_{bk}$ are assumed to be learned *a priori* from the entire sequence and supplied as constants in the motif detection process.) The symbols $\Theta_g$, $\Theta_l$, $\mathcal{M}_g$, $\mathcal{M}_l$ stand for the parameters and model classes in the respective submodels. Thus, the likelihood of a regulatory sequence $y$ is:

$$p(y|\Theta, \mathcal{M}) = \sum_x p(x|\Theta_g, \mathcal{M}_g)p(y|x, \Theta_l, \mathcal{M}_l)$$

$$= \sum_x p(x|\Theta_g, \mathcal{M}_g)p(\mathbf{A}|x, \Theta_l, \mathcal{M}_l)p(y - \mathbf{A}|x, \Theta_{bk}), \qquad (1)$$

where $\mathbf{A} \triangleq \mathbf{A}(x,y)$. Note that $\Theta_l$ here is not necessarily equivalent to the position-specific multinomial parameters $\Theta$ in Eq. 2 below, but is a generic symbol for the parameters of a general model of aligned motif instances.

The model $p(x|\Theta_g, \mathcal{M}_g)$ captures properties such as the frequencies of different motifs and the dependencies between motif occurrences. Although specifying this model is an important aspect of motif detection and remains largely unexplored, we defer this issue to future work. In the current paper, our focus is on capturing the intrinsic properties within motifs that can help to improve sensitivity and specificity to genuine motif patterns. For this the key lies in the local alignment model $p(A(x,y)|x, \Theta_l, \mathcal{M}_l)$, which determines the PWM of the motif. Depending on the value of the latent indicator $x_t$ (a motif or not at position $t$), $y_t$ admits different probabilistic models, such as a motif alignment model or a background model. Thus sequence $y$ is characterized by a *Bayesian multinet* [6], a mixture model in which each component of the mixture is a specific nt-distribution model corresponding to sequences of a particular nature. Our goal in this paper is to develop an expressive local alignment model $p(A(x,y)|x, \Theta_l, \mathcal{M}_l)$ capable of capturing characteristic site-dependencies in motifs.

In the standard product-multinomial (PM) model for local alignment, the columns of a PWM are assumed to be independent [9]. Thus the likelihood of a multi-alignment $\mathbf{A}$ is:

$$p(\mathbf{A}|\Theta) = \prod_{l=1}^{L} \prod_{j=1}^{4} \left[\theta_{lj}\right]^{h_{lj}}. \qquad (2)$$

Although a popular model for many motif finders, PM nevertheless is sensitive to noise and random or trivial recurrent patterns, and is unable to capture potential site-dependencies inside the motifs. Pattern-driven auxiliary submodels (e.g., the fragmentation model [10]) or heuristics (e.g., split a 'two-block' motif into two coupled sub-motifs [9, 1]) have been developed to handle special patterns such as the *U-shaped* motifs, but they are inflexible and difficult to generalize. Some of the literature has introduced vague Dirichlet priors for $\theta$ in the PM [2, 10], but they are primarily used for smoothing rather than for explicitly incorporating prior knowledges about motifs.

We depart from the PM model and introduce a dynamic hierarchical Bayesian model for motif alignment $\mathbf{A}$, which captures site dependencies inside the motif so that we can predict biologically more plausible motifs, and incorporate prior knowledge of nucleotide frequencies of general motif sites. In order to keep the local alignment model our main focus as well as simplifying the presentation, we adopt an idealized global motif distribution model called "one-per-sequence" [8], which, as the name suggests, assumes each sequence harbors one motif instance (at an unknown location). Generalization to more expressive global models is straightforward and is described in the full paper.

### 3.2 Hidden Markov Dirichlet-Multinomial (HMDM) Model

In the HMDM model, we assume that there are $K$ underlying latent nt-distribution proto-types, according to which position-specific multinomial distributions of nt are determined, and that each prototype is represented by a Dirichlet distribution. Furthermore, the choice of prototype at each position in the motif is governed by a first-order Markov process.

More precisely, a multi-alignment $\mathbf{A}_{M \times L}$ containing $M$ motif instances is generated by the following process. First we sample a sequence of prototype indicators $q = (q_1, \ldots, q_L)$ from a first-order Markov process with initial distribution $\pi$ and transition matrix $B$. Then we repeat the following for each column $l \in \{1, \ldots, L\}$: (1) A component from a mixture of Dirichlets $\alpha = \{\vec{\alpha}_1, \ldots, \vec{\alpha}_K\}$, where each $\vec{\alpha}_i = (\alpha_{i,1}, \ldots, \alpha_{i,4})$, is picked according to indicator $q_l$. Say we picked $\vec{\alpha}_i$. (2) A multinomial distribution $\vec{\theta}_l$ is sampled according to $p(\vec{\theta}|\vec{\alpha}_i)$, the probability defined by Dirichlet component $i$ over all such distributions. (3) All the nucleotides in column $l$ are generated i.i.d. according to Multi($\vec{\theta}_l$).

The complete likelihood of motif alignment $\mathbf{A}_{M \times L}$ characterized by counting matrix $h$ is:

$$p(\mathbf{A}, q, \theta|\Theta_l, \mathcal{M}_l) = p(h|\theta)p(\theta|q, \alpha)p(q|\pi, B)$$

$$= \prod_{l=1}^{L}\prod_{i=1}^{K} C(\alpha_i)^{q_l^i} \prod_{j=1}^{4}[\theta_{l,j}]^{\sum_{i=1}^{K} q_l^i (\alpha_{i,j}+h_l^j-1)} \prod_{i=1}^{K}[\pi_i]^{q_1^i} \prod_{l=1}^{L-1}\prod_{i,j=1}^{K}[B_{i,j}]^{q_l^i q_{l+1}^j} \quad (3)$$

The major role of HMDM is to impose dynamic priors for modeling data whose distributions exhibit temporal or spatial dependencies. As Figure 2(b) makes clear, this model is not a simple HMM for discrete sequences. In such a model the transition would be between the emission models (i.e., multinomials) themselves, and the output at each time would be a single data instance in the sequence. In HMDM, the transitions are between different priors of the emission models, and the direct output of the HMM is the parameter vector of a generative model, which will be sampled multiple times at each position to generate random instances.

This approach is especially useful when we have either empirical or learned prior knowl-edge about the dynamics of the data to be modeled. For example, for the case of motifs, bi-ological evidence show that conserved positions (manifested by a low-entropy multinomial nt-distribution) are likely to concatenate, and maybe so do the less conserved positions. However, it is unlikely that conserved and less conserved positions are interpolated [4]. This is called *site clustering*, and is one of the main motivations for the HMDM model.

## 4 Inference and Learning

### 4.1 Variational Bayesian Learning

In order to do Bayesian estimation of the motif parameter $\theta$, and to predict the locations of motif instances via $x$, we need to be able to compute the posterior distribution $p(\theta|y)$, which is infeasible in a complex motif model. Thus we turn to variational approximation [5]. We seek to approximate the joint posterior over parameters and hidden states $p(\theta, q, x|y, \mathcal{M})$ with a simpler distribution $Q(\theta, q, x) = Q_{\theta,q}(\theta, q)Q_x(x)$, where $Q_{\theta,q}$ and $Q_{\theta,q}$ can be, for the time being, thought of as free distributions to be optimized. Using Jensen's inequality, we have the following lower found on the log likelihood:

$$\ln p(y|\mathcal{M}) \geq \int d\theta dq Q_{\theta,q}(\theta, q) \left[ \int dx Q_x(x) \ln \frac{p(y, x|\theta, \mathcal{M})}{Q_x(x)} + \ln \frac{p(\theta, q|\mathcal{M})}{Q_{\theta,q}(\theta, q)} \right] (4)$$

$$= -\text{KL}(Q(\theta, q, x)\|p(\theta, q, x|y, \mathcal{M})) + C.$$

Thus, maximizing the lower bound of the log likelihood (call it $\mathcal{B}(Q_x, Q_{\theta,q})$) with respect to free distributions $Q_x$ and $Q_{\theta,q}$ is equivalent to minimizing the KL divergence between the true joint posterior and its variational approximation. Keeping either $Q_x$ or $Q_{\theta,q}$ fixed

and maximizing $\mathcal{B}(\cdot)$ with respect to the other, we obtain the following coupled updates:

$$Q_x^*(x) \quad \propto \quad \exp \langle \ln p(y, x | \theta, \mathcal{M}) \rangle_{Q_{\theta,q}} \tag{5}$$

$$Q_{\theta,q}^*(\theta, q) \quad \propto \quad p(\theta, q | \mathcal{M}) \exp \langle \ln p(y, x | \theta, \mathcal{M}) \rangle_{Q_x} \tag{6}$$

In our motif model, the prior and the conditional submodels form a *conjugate-exponential* pair (Dirichlet-Multinomial). It can be shown that in this case we can essentially recover the same form of the original conditional and prior distributions in their variational approximations except that the parameterization is augmented with appropriate Bayesian and posterior updates, respectively:

$$Q_x(x) \quad = \quad p(x, y | \bar{\phi}(\theta), \mathcal{M}) = p(y | x, \bar{\phi}(\theta), \mathcal{M}) p(x | \mathcal{M}) \tag{7}$$

$$Q_{\theta,q}(\theta, q) \quad = \quad p(\theta, q | \alpha, \bar{h}(x, y), \mathcal{M}) \tag{8}$$

where $\bar{\phi}(\theta) = \langle \phi(\theta) \rangle_{Q_{\theta,q}}$ ($\phi(\cdot)$ is the natural parameter) and $\bar{h}(x, y) = \langle h(x, y) \rangle_{Q_x}$.

As Eqs. 7 and 8 make clear, the locality of inference and marginalization on the latent variables is preserved in the variational approximation, which means probabilistic calculations can be performed in the prior and the conditional models separately and iteratively. For motif modeling, this modular property means that the motif alignment model and motif distribution model can be treated separately with a simple interface of the posterior mean for the motif parameters and expected sufficient statistics for the motif instances.

## 4.2 Inference and learning

According to Eq. 8, we replace the counting matrix $h$ in Eq. 3, which is the output of the HMDM model, by the expected counting matrix $\langle h \rangle$ obtained from inference in the global distribution model (we will handle this later, thanks to the locality preservation property of inference in variational approximations), and proceed with the inference as if we have "observations" $\langle h \rangle$. Integrating over $\theta$, we have the marginal distribution:

$$p(q, \langle h \rangle) \quad = \quad p(q_1) \prod_{l=1}^{L-1} p(q_{l+1} | q_l) \prod_{l=1}^{L} p(\langle h_l \rangle | q_l), \tag{9}$$

a standard HMM with emission probability:

$$p(\langle h_l \rangle | q_l^i = 1) \quad = \quad \frac{\Gamma(|\vec{\alpha}_i|)}{\Gamma(|\langle h_l \rangle| + |\vec{\alpha}_i|)} \prod_{j=1}^{4} \frac{\Gamma(\langle h_l^j \rangle + \alpha_{i,j})}{\Gamma(\alpha_{i,j})}. \tag{10}$$

We can compute the posterior probability of the hidden states $p(q_l | \langle h \rangle)$ and the matrix of co-occurrence probabilities $p(q_l, q_{l+1} | \langle h \rangle)$ using standard forward-backward algorithm.

We next compute the expectation of the natural parameters (which is $\ln \theta$ for multinomial parameters). Given the "observations" $\langle h \rangle$, the posterior mean is computed as follows:

$$\bar{\phi}(\theta_{l,j}) \quad = \quad \int_\theta \sum_{q_l} \ln \theta_{l,j} p(\vec{\theta}_l, q_l | \alpha, \langle h \rangle, \mathcal{M}) d\vec{\theta}_l$$

$$= \quad \sum_{i=1}^{K} \gamma(q_l^i) \left( \Psi(\alpha_{i,j} + \langle h_l^j \rangle) - \Psi(|\vec{\alpha}_i| + |\langle h_l \rangle|) \right), \tag{11}$$

where $\gamma(q_l)$ is the posterior probability of the hidden state (an output of the forward-backward algorithm) and $\Psi(x) = \frac{\partial \log \Gamma(x)}{\partial x} = \frac{\Gamma'(x)}{\Gamma(x)}$ is the digamma function.

Following Eq. 7, given the posterior means of the multinomial parameters, computing the expected counting matrix $\langle h \rangle$ under the the one-per-sequence global model for sequence set $\{y^{(1)}, \ldots, y^{(M)}\}$ is straightforward based on Eq. 2 and we simply give the final results:

$$\langle h_{l,j} \rangle = \frac{1}{N} \sum_{n=1}^{N} \sum_{t=1}^{T_n - L + 1} Q^{(n)}(t) \delta(y_{t+l-1}^{(n)}, j), \tag{12}$$

$$\text{where} \quad Q^{(n)}(t) \propto \prod_{l=0}^{L-1} \prod_{j=1}^{4} \left[ \frac{\bar{\theta}_{l,j}}{\theta_{0,j}} \right]^{\delta(y_{t+l}^{(n)}, j)} = \exp \left\{ \sum_{l=0}^{L-1} \sum_{j=1}^{4} \delta(y_{t+l}^{(n)}, j) \left( \bar{\phi}(\theta_{l,j}) - \phi(\theta_{0,j}) \right) \right\}$$
(13)

Bayesian estimates of the multinomial parameters for the position-specific nt-distribution of the motif are obtained via fixed-point iteration under the following EM-like procedure:

- **Variational E step**: Compute the expected sufficient statistic, the count matrix $\langle h \rangle$, via inference in the global motif model given $\bar{\phi}(\theta_{l,j})$.

- **Variational M step**: Compute the expected natural parameter $\bar{\phi}(\theta_{l,j})$ via inference in the local motif alignment model given $\langle h \rangle$.

This basic inference and learning procedure provides a framework that scales readily to more complex models. For example, the motif distribution model $p(x)$ can be made more sophisticated so as to model complex properties of multiple motifs such as motif-level dependencies (e.g., co-occurrence, overlaps and concentration within regulatory modules) without complicating the inference in the local alignment model. Similarly, the motif alignment model can also be more expressive (e.g., a mixture of HMDMs) without interfering with inference in the motif distribution model.

## 5   Experiments

We test the HMDM model on a motif collection from *The Promoter Database of Saccharomyces cerevisiae* (SCPD). Our dataset contains twenty motifs, each has 6 to 32 instances all of which are identified via biological experiments.

We begin with an experiment showing how HMDM can capture intrinsic properties of the motifs. The posterior distribution of the position-specific multinomial parameters $\theta$, reflected in the parameters of the Dirichlet mixtures learned from data, can reveal the nt-distribution patterns of the motifs. Examining the transition probabilities between different Dirichlet components further tells us the about dependencies between adjacent positions (which indirectly reveals the "shape" information). We set the total number of Dirichlet components to be 8 based on an intelligent guess (using biological intuition), and Figure 3(a) shows the Dirichlet parameters fitted from the dataset via empirical Bayes estimation. Among the 8 Dirichlet components, numbers 1-4 favor a pure distribution of single nucleotides A, T, G, C, respectively, suggesting they correspond to "homogeneous" prototypes. Whereas numbers 7 and 8 favor a near uniform distribution of all 4 nt-types, hence "heterogeneous" prototypes. Components 5 and 6 are somewhat in between. Such patterns agree well with the biological definition of motifs. Interestingly, from the learned transition model of the HMM (Figure 3(b)), it can be seen that the transition probability from a homogeneous prototype to a heterogeneous prototype is significantly less than that between two homogeneous or two heterogeneous prototypes, confirming an empirical speculation in biology that motifs have the so-called *site clustering* property [4].

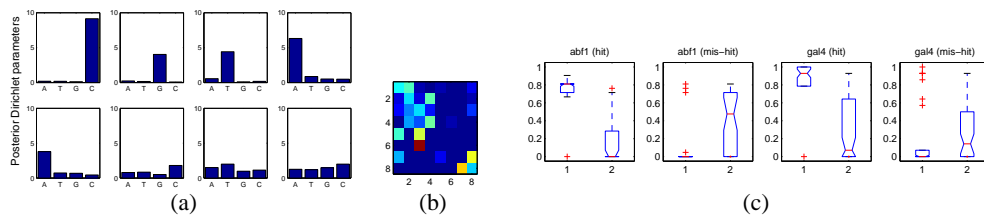

Figure 3: (a) Dirichlet hyperparameters. (b) Markov transition matrix. (c) Boxplots of hit and mishit rate of HMDM(1) and PM(2) on two motifs used during HMDM training.

Are the motif properties captured in HMDM useful in motif detection? We first examine an HMDM trained on the complete dataset for its ability to detect motifs used in training in the presence of a "decoy": a permuted motif. By randomly permuting the positions in the

motif, the shapes of the "U-shaped" motifs (e.g., *abf1* and *gal4*) change dramatically.[2] We insert each instance of motif/decoy pair into a 300-500 bp random background sequence at random position $b$ and $b'$.[3] We allow a $\pm 3$ bp offset as a tolerance window, and score a *hit* when $b - 3 \le a \le b + 3$ (and a *mis-hit* when $b' - 3 \le a \le b' + 3$), where $a$ is the position where a motif instance is found. The (mis)hit rate is the proportion of (mis)hits to the total number of motif instances to be found in an experiment. Figure 3(c) shows a boxplot of the hit and mishit rate of HMDM on *abf1* and *gal4* over 50 randomly generated experiments. Note the dramatic contrast of the sensitivity of the HMDM to true motifs compared to that of the PM model (which is essentially the MEME model).

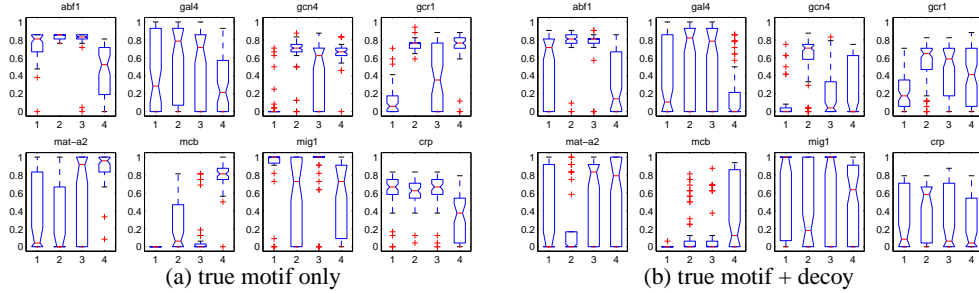

Figure 4: Motif detection on an independent test dataset (the 8 motifs in Figure 1(a)). Four models used are indexed as: 1. HMDM(bell); 2. HMDM(U); 3. HMDM-mixture; 4. PM. Boxplot of hit-rate is for 80 randomly generated experiments (the center of the notch is the median).

How well does HMDM generalize? We split our data into a training set and a testing set, and further divide the training set roughly based on bell-shaped and U-shaped patterns to train two different HMDMs, respectively, and a mixture of HMDMs. In the first motif finding task, we are given sequences each of which has only one true motif instance at a random position. The results are given in Figure 4(a). We see that for 4 motifs, using an HMDM or the HMDM-mixtures significantly improves performance over PM model. In three other cases they are comparable, but for motif *mcb*, all HMDM models lose. Note that *mcb* is very "conserved," which is in fact "atypical" in the training set. It is also very short, which diminishes the utility of an HMM. Another interesting observation from Figure 4(a) is that even when both HMDMs perform poorly, the HMDM-mixtures can still perform well (e.g., *mat-a2*), presumably because of the extra flexibility provided by the mixture model.

The second task is more challenging and biologically more realistic, where we have both the true motifs and the permuted "decoys." We show only the hit-rate over 80 experiments in Figure 4(b). Again, in most cases HMDM or the HMDM mixture outperforms PM.

## 6 Conclusions

We have presented a generative probabilistic framework for modeling motifs in biopolymer sequences. Naively, categorical random variables with spatial/temporal dependencies can be modeled by a standard HMM with multinomial emission models. However, the limited flexibility of each multinomial distribution and the concomitant need for a potentially large number of states to model complex domains may require a large parameter count and lead to overfitting. The infinite HMM [3] solve this issue by replacing the emission model with a Dirichlet process which provides potentially infinite flexibility. However, this approach is purely data-driven and provides no mechanism for explicitly capturing multi-modality

in the emission and the transition models or for incorporating informative priors. Furthermore, when the output of the HMM involves hidden variables (as for the case of motif detection), inference and learning is further complicated.

HMDM assumes that positional dependencies are induced at a higher level among the finite number of informative Dirichlet priors rather than between the multinomials themselves. Within such a framework, we can explicitly capture the multi-modalities of the multinomial distributions governing the categorical variable (such as motif sequences at different positions) and the dependencies between modalities, by learning the model parameters from training data and using them for future predictions. In motif modeling, such a strategy was used to capture different distribution patterns of nucleotides (*homogeneous* and *heterogeneous*) and transition properties between patterns (*site clustering*). Such a prior proves to be beneficial in searching for unseen motifs in our experiment and helps to distinguish more probable motifs from biologically meaningless random recurrent patterns.

Although in the motif detection setting the HMDM model involves a complex missing data problem in which both the output and the internal states of the HMDM are hidden, we show that a variational Bayesian learning procedure allows probabilistic inference in the prior model of motif sequence patterns and in the global distribution model of motif locations to be carried out virtually separately with a Bayesian interface connecting the two processes. This divide and conquer strategy makes it much easier to develop more sophisticated models for various aspects of motif analysis without being overburdened by the somewhat daunting complexity of the full motif problem.

## Footnotes

[1] Multiple motif detection can be formulated in a similar way, but for simplicity, we omit this elaboration. See full paper for details. Also for simplicity, we omit the superscript $n$ (sequence index) of variable $x$ and $y$ in wherever it is unnecessary.

[2]By permutation we mean each time the same permuted order is applied to all the instances of a motif so that the multinomial distribution of each position is not changed but their order changed.

[3]We resisted the temptation of using biological background sequences because we would not know if and how many other motifs are in such sequences, which renders them ill-suited for purposes of evaluation.

## References

[1] T. L. Bailey and C. Elkan. Unsupervised learning of multiple motifs in biopolymers using EM. *Machine Learning*, 21:51–80, 1995.

[2] T. L. Bailey and C. Elkan. The value of prior knowledge in discovering motifs with MEME. In *Proc. of the 3rd International Conf. on Intelligent Systems for Molecular Biology*, 1995.

[3] M. J. Beal, Z. Ghahramani, and C. E. Rasmussen. The infinite hidden Markov model. In *Proc. of 14th Conference on Advances in Neural Information Processing Systems*, 2001.

[4] M. Eisen. Structural properties of transcription factor-DNA interactions and the inference of sequence specificity. manuscript in preparation.

[5] Z. Ghahramani and M.J. Beal. Propagation algorithms for variational Bayesian learning. In *Proc. of 13th Conference on Advances in Neural Information Processing Systems*, 2000.

[6] D. Heckerman, D. Geiger, and D. M. Chickering. Learning Bayesian networks: the combination of knowledge and statistics data. *Machine Learning*, 20:197–243, 1995.

[7] C. Lawrence and A. Reilly. An expectation maximization (EM) algorithm for the identification and characterization of common sites in unaligned biopolymer sequences. *Proteins*, 7:41–51, 1990.

[8] C.E. Lawrence, S.F. Altschul, M.S. Boguski, J.S. Liu, A.F. Neuwald, and J.C. Wootton. Detecting subtle sequence signals: A Gibbs sampling strategy for multiple alignment. *Science*, 262:208–214, 1993.

[9] J. Liu, X. Liu, and D.L. Brutlag. Bioprospector: Discovering conserved DNA motifs in upstream regulatory regions of co-expressed genes. In *Proc. of PSB*, 2001.

[10] J.S. Liu, A.F. Neuwald, and C.E. Lawrence. Bayesian models for multiple local sequence alignment and Gibbs sampling strategies. *J. Amer. Statistical Assoc*, 90:1156–1169, 1995.

[11] A. M. Michelson. Deciphering genetic regulatory codes: A challenge for functional genomics. *Proc. Natl. Acad. Sci. USA*, 99:546–548, 2002.
